# A harmonic excitation state-space approach to blind separation of speech

**Rasmus Kongsgaard Olsson and Lars Kai Hansen**
Informatics and Mathematical Modelling
Technical University of Denmark, 2800 Lyngby, Denmark
`rko,lkh@imm.dtu.dk`

## Abstract

We discuss an identification framework for noisy speech mixtures. A block-based generative model is formulated that explicitly incorporates the time-varying harmonic plus noise (H+N) model for a number of latent sources observed through noisy convolutive mixtures. All parameters including the pitches of the source signals, the amplitudes and phases of the sources, the mixing filters and the noise statistics are estimated by maximum likelihood, using an EM-algorithm. Exact averaging over the hidden sources is obtained using the Kalman smoother. We show that pitch estimation and source separation can be performed simultaneously. The pitch estimates are compared to laryngograph (EGG) measurements. Artificial and real room mixtures are used to demonstrate the viability of the approach. Intelligible speech signals are re-synthesized from the estimated H+N models.

## 1 Introduction

Our aim is to understand the properties of mixtures of speech signals within a generative statistical framework. We consider *convolutive* mixtures, i.e.,

$$\mathbf{x}_t = \sum_{k=0}^{L-1} \mathbf{A}_k \mathbf{s}_{t-k} + \mathbf{n}_t, \tag{1}$$

where the elements of the source signal vector, $\mathbf{s}_t$, i.e., the $d_s$ statistically independent source signals, are convolved with the corresponding elements of the filter matrix, $\mathbf{A}_k$. The multichannel sensor signal, $\mathbf{x}_t$, is furthermore degraded by additive Gaussian white noise.

It is well-known that separation of the source signals based on second order statistics is infeasible in general. Consider the second order statistic

$$\langle \mathbf{x}_t \mathbf{x}_{t'}^\top \rangle = \sum_{k,k'=0}^{L-1} \mathbf{A}_k \langle \mathbf{s}_{t-k} \mathbf{s}_{t'-k'}^\top \rangle \mathbf{A}_{k'}^\top + \mathbf{R}, \tag{2}$$

where $\mathbf{R}$ is the (diagonal) noise covariance matrix. If the sources can be assumed stationary white noise, the source covariance matrix can be assumed proportional to the unit matrix

without loss of generality, and we see that the statistic is symmetric to a common rotation of all mixing matrices $\mathbf{A}_k \rightarrow \mathbf{A}_k\mathbf{U}$. This rotational invariance means that the acquired statistic is not informative enough to identify the mixing matrix, hence, the source time series.

However, if we consider stationary sources with *known*, non-trivial, autocorrelations $\langle \mathbf{s}_t\mathbf{s}_{t'}^\top \rangle = \mathbf{G}(t - t')$, and we are given access to measurements involving multiple values of $\mathbf{G}(t - t')$, the rotational degrees of freedom are constrained and we will be able to recover the mixing matrices up to a choice of sign and scale of each source time series. Extending this argument by the observation that the mixing model (1) is invariant to filtering of a given column of the convolutive filter provided that the inverse filter is applied to corresponding source signal, we see that it is infeasible to identify the mixing matrices if these arbitrary inverse filters can be chosen to that they are allowed to 'whiten' the sources, see also [1].

*For non-stationary sources, on the other hand, the autocorrelation functions vary through time and it is not possible to choose a single common whitening filter for each source.* This means that the mixing matrices may be identifiable from multiple estimates of the second order correlation statistic (2) for non-stationary sources. Analysis in terms of the number of free parameters vs. the number of linear conditions is provided in [1] and [2].

Also in [2], the constraining effect of source non-stationarity was exploited by the simultaneous diagonalization of multiple estimates of the source power spectrum. In [3] we formulated a generative probabilistic model of this process and proved that it could estimate sources and mixing matrices in noisy mixtures. Blind source separation based on state-space models has been studied, e.g., in [4] and [5]. The approach is especially useful for including prior knowledge about the source signals and for handling noisy mixtures. One example of considerable practical importance is the case of speech mixtures.

For speech mixtures the generative model based on white noise excitation may be improved using more realistic priors. Speech models based on *sinusoidal* excitation have been quite popular in speech modelling since [6]. This approach assumes that the speech signal is a time-varying mixture of a harmonic signal and a noise signal (H+N model). A recent application of this model for pitch estimation can be found in [7]. Also [8] and [9] exploit the harmonic structure of certain classes of signals for enhancement purposes. A related application is the BSS algorithm of [10], which uses the cross-correlation of the amplitude in different frequency. The state-space model naturally leads to maximum-likelihood estimation using the EM-algorithm, e.g. [11], [12]. The EM algorithm has been used in related models: [13] and [14].

*In this work we generalize our previous work on state space models for blind source separation to include harmonic excitation and demonstrate that it is possible to perform simultaneous un-mixing and pitch tracking.*

## 2   The model

The assumption of time variant source statistics help identify parameters that would otherwise not be unique within the model. In the following, the measured signals are *segmented* into frames, in which they are assumed stationary. The mixing filters and observation noise covariance matrix are assumed stationary across *all* frames.

The colored noise (AR) process that was used in [3] to model the sources is augmented to include a periodic excitation signal that is also time-varying. The specific choice of periodic basis function, i.e. the sinusoid, is motivated by the fact that the phase is linearizable,

facilitating one-step optimization. In frame $n$, source $i$ is represented by:

$$
\begin{aligned}
s_{i,t}^n &= \sum_{t'=1}^{p} f_{i,t'}^n s_{i,t-t'}^n + \sum_{k=1}^{K} \alpha_{i,k}^n \sin(\omega_{0,i}^n kt + \beta_i^n) + v_{i,t}^n \\
&= \sum_{t'=1}^{p} f_{i,t'}^n s_{i,t-t'}^n + \sum_{k=1}^{K} c_{i,2k-1}^n \sin(\omega_{0,i}^n kt) + c_{i,2k}^n \cos(\omega_{0,i}^n kt) + v_{i,t}^n \quad (3)
\end{aligned}
$$

where $n \in \{1,2,..,N\}$ and $i \in \{1,2,..,d_s\}$. The innovation noise, $v_{i,t}^n$, is i.i.d Gaussian. Clearly, (3) represents a H+N model. The fundamental frequency, $\omega_{0,i}^n$, enters the estimation problem in an inherent non-linear manner.

In order to benefit from well-established estimation theory, the above recursion is fitted into the framework of Gaussian linear models, see [15]. The Kalman filter model is an instance of this model. The augmented state space is constructed by including a history of past samples for each source. Source vector $i$ in frame $n$ is defined: $\mathbf{s}_{i,t}^n = \begin{bmatrix} s_{i,t}^n & s_{i,t-1}^n & \cdots & s_{i,t-p+1}^n \end{bmatrix}^\top$. All $\mathbf{s}_{i,t}^n$'s are stacked in the total source vector: $\bar{\mathbf{s}}_t^n = \begin{bmatrix} (\mathbf{s}_{1,t}^n)^\top & (\mathbf{s}_{2,t}^n)^\top & \cdots & (\mathbf{s}_{d_s,t}^n)^\top \end{bmatrix}^\top$. The resulting state-space model is:

$$
\begin{aligned}
\bar{\mathbf{s}}_t^n &= \mathbf{F}^n \bar{\mathbf{s}}_{t-1}^n + \mathbf{C}^n \mathbf{u}_t^n + \bar{\mathbf{v}}_t^n \\
\mathbf{x}_t^n &= \mathbf{A} \bar{\mathbf{s}}_t^n + \mathbf{n}_t^n
\end{aligned}
$$

where $\bar{\mathbf{v}}_t \sim \mathcal{N}(\mathbf{0}, \mathbf{Q})$, $\mathbf{n}_t \sim \mathcal{N}(\mathbf{0}, \mathbf{R})$ and $\bar{\mathbf{s}}_1^n \sim \mathcal{N}(\mu^n, \mathbf{\Sigma}^n)$. The combined harmonics input vector is defined: $\mathbf{u}_t^n = \begin{bmatrix} (\mathbf{u}_{1,t}^n)^\top & (\mathbf{u}_{2,t}^n)^\top & \cdots & (\mathbf{u}_{d_s,t}^n)^\top \end{bmatrix}^\top$, where the harmonics corresponding to source $i$ in frame $n$ are:

$$
\mathbf{u}_{i,t}^n = \begin{bmatrix} \sin(\omega_{0,i}^n t) & \cos(\omega_{0,i}^n t) & \cdots & \sin(K\omega_{0,i}^n t) & \cos(K\omega_{0,i}^n t) \end{bmatrix}^\top
$$

It is apparent that the matrix multiplication by $\mathbf{A}$ constitutes a *convolutive* mixing of the sources, where the $d_x \times d_s$ channel filters are:

$$
\mathbf{A} = \begin{bmatrix}
\mathbf{a}_{11}^\top & \mathbf{a}_{12}^\top & .. & \mathbf{a}_{1d_s}^\top \\
\mathbf{a}_{21}^\top & \mathbf{a}_{22}^\top & .. & \mathbf{a}_{2d_s}^\top \\
\vdots & \vdots & \ddots & \vdots \\
\mathbf{a}_{d_x 1}^\top & \mathbf{a}_{d_x 2}^\top & .. & \mathbf{a}_{d_x d_s}^\top
\end{bmatrix}
$$

In order to implement the H+N source model, the parameter matrices are constrained as follows:

$$
\mathbf{F}^n = \begin{bmatrix}
\mathbf{F}_1^n & \mathbf{0} & \cdots & \mathbf{0} \\
\mathbf{0} & \mathbf{F}_2^n & \cdots & \mathbf{0} \\
\vdots & \vdots & \ddots & \vdots \\
\mathbf{0} & \mathbf{0} & \cdots & \mathbf{F}_{d_s}^n
\end{bmatrix}
\,,\quad
\mathbf{F}_i^n = \begin{bmatrix}
f_{i,1}^n & f_{i,2}^n & \cdots & f_{i,p-1}^n & f_{i,p}^n \\
1 & 0 & \cdots & 0 & 0 \\
0 & 1 & \cdots & 0 & 0 \\
\vdots & \vdots & \ddots & \vdots & \vdots \\
0 & 0 & \cdots & 1 & 0
\end{bmatrix}
$$

$$
\mathbf{Q}^n = \begin{bmatrix}
\mathbf{Q}_1^n & \mathbf{0} & \cdots & \mathbf{0} \\
\mathbf{0} & \mathbf{Q}_2^n & \cdots & \mathbf{0} \\
\vdots & \vdots & \ddots & \vdots \\
\mathbf{0} & \mathbf{0} & \cdots & \mathbf{Q}_{d_s}^n
\end{bmatrix}
\,,\quad
(\mathbf{Q}_i^n)_{jj'} = \begin{cases} q_i^n & j = j' = 1 \\ 0 & j \neq 1 \bigvee j' \neq 1 \end{cases}
$$

$$
\mathbf{C}^n = \begin{bmatrix}
\mathbf{C}_1^n & \mathbf{0} & \cdots & \mathbf{0} \\
\mathbf{0} & \mathbf{C}_2^n & \cdots & \mathbf{0} \\
\vdots & \vdots & \ddots & \vdots \\
\mathbf{0} & \mathbf{0} & \cdots & \mathbf{C}_{d_s}^n
\end{bmatrix}
\,,\quad
\mathbf{C}_i^n = \begin{bmatrix}
c_{i,1}^n & c_{i,2}^n & \cdots & c_{i,2K}^n \\
0 & 0 & \cdots & 0 \\
0 & 0 & \cdots & 0 \\
\vdots & \vdots & \ddots & \vdots \\
0 & 0 & \cdots & 0
\end{bmatrix}
$$

## 3  Learning

Having described the convolutive mixing problem in the general framework of linear Gaussian models, more specifically the Kalman filter model, optimal inference of the sources is obtained by the Kalman smoother. However, since the problem at hand is effectively *blind*, we also need to estimate the parameters. Along the lines of, e.g. [15], we will invoke an EM approach. The log-likelihood is bounded from below: $\mathcal{L}(\theta) \geq \mathcal{F}(\theta, \hat{p}) \equiv \mathcal{J}(\theta, \hat{p}) - \mathcal{R}(\hat{p})$, with the definitions $\mathcal{J}(\theta, \hat{p}) \equiv \int d\mathbf{S}\hat{p}(\mathbf{S}) \log p(\mathbf{X}, \mathbf{S}|\theta)$ and $\mathcal{R}(\hat{p}) \equiv \int d\mathbf{S}\hat{p}(\mathbf{S}) \log \hat{p}(\mathbf{S})$. In accordance with standard EM theory, $\mathcal{J}(\theta, \hat{p})$ is optimized wrt. $\theta$ in the M-step. The E-step infers the relevant moments of the marginal posterior, $\hat{p} = p(\mathbf{S}|\mathbf{X}, \theta)$. For the Gaussian model the means are also source MAP estimates. The combined E and M steps are guaranteed not to decrease $\mathcal{L}(\theta)$.

### 3.1  E-step

The forward-backward recursions which comprise the Kalman smoother are employed in the E-step to infer moments of the source posterior, $p(\mathbf{S}|\mathbf{X}, \theta)$, i.e. the joint posterior of the sources conditioned on all observations. The relevant second-order statistic of this distribution in segment $n$ is the marginal posterior mean, $\hat{\bar{\mathbf{s}}}_t^n \equiv \langle \bar{\mathbf{s}}_t^n \rangle$, and autocorrelation, $\mathbf{M}_{i,t}^n \equiv \langle \mathbf{s}_{i,t}^n (\mathbf{s}_{i,t}^n)^\top \rangle \equiv [ \ \mathbf{m}_{i,1,t}^n \quad \mathbf{m}_{i,2,t}^n \quad .. \quad \mathbf{m}_{i,L,t}^n \ ]^\top$, along with the marginal lag-one covariance, $\mathbf{M}_{i,t}^{1,n} \equiv \langle \mathbf{s}_{i,t}^n (\mathbf{s}_{i,t-1}^n)^\top \rangle \equiv [ \ \mathbf{m}_{i,1,t}^{1,n} \quad \mathbf{m}_{i,2,t}^{1,n} \quad .. \quad \mathbf{m}_{i,L,t}^{1,n} \ ]^\top$. In particular, $m_{i,t}^n$ is the first element of $\mathbf{m}_{i,1,t}^n$. All averages are performed over $p(\mathbf{S}|\mathbf{X}, \theta)$. The forward recursion also yields the log-likelihood, $\mathcal{L}(\theta)$.

### 3.2  M-step

The M-step utility function, $\mathcal{J}(\theta, \hat{p})$, is defined:

$$\mathcal{J}(\theta, \hat{p}) = -\frac{1}{2} \sum_{n=1}^{N} [\sum_{i=1}^{d_s} \log \det \mathbf{\Sigma}_i^n + (\tau - 1) \sum_{i=1}^{d_s} \log q_i^n$$

$$+ \tau \log \det \mathbf{R} + \sum_{i=1}^{d_s} \langle (\mathbf{s}_{i,1}^n - \mathbf{\mu}_i^n)^T (\mathbf{\Sigma}_i^n)^{-1} (\mathbf{s}_{i,1}^n - \mathbf{\mu}_i^n) \rangle$$

$$+ \sum_{t=2}^{\tau} \sum_{i=1}^{d_s} \langle \frac{1}{q_i^n} (s_{i,t}^n - (\mathbf{d}_i^n)^T \mathbf{z}_{i,t}^n)^2 \rangle + \sum_{t=1}^{\tau} \langle (\mathbf{x}_t^n - \mathbf{A}\bar{\mathbf{s}}_t^n)^T \mathbf{R}^{-1} (\mathbf{x}_t^n - \mathbf{A}\bar{\mathbf{s}}_t^n) \rangle ]$$

where $\langle \cdot \rangle$ signifies averaging over the source posterior from the previous E-step, $p(\mathbf{S}|\mathbf{X}, \theta)$ and $\tau$ is the frame length. The linear source parameters are grouped as

$$\mathbf{d}_i^n \equiv [ \ (\mathbf{f}_i^n)^\top \quad (\mathbf{c}_i^n)^\top \ ]^\top \quad , \quad \mathbf{z}_i^n \equiv [ \ (\mathbf{s}_{i,t-1}^n)^\top \quad (\mathbf{u}_{i,t}^n)^\top \ ]^\top$$

where

$$\mathbf{f}_i^n \equiv [ \ f_{i,1} \quad f_{i,2} \quad .. \quad f_{i,p} \ ]^\top \quad , \quad \mathbf{c}_i^n \equiv [ \ c_{i,1} \quad c_{i,2} \quad .. \quad c_{i,p} \ ]^\top$$

Optimization of $\mathcal{J}(\theta, \hat{p})$ wrt. $\theta$ is straightforward (except for the $\omega_{0,i}^n$'s). Relatively minor changes are introduced to the estimators of e.g. [12] in order to respect the special constrained format of the parameter matrices and to allow for an external input to the model. More details on the estimators for the correlated source model are given in [3].

It is in general difficult to maximize $\mathcal{J}(\theta, \hat{p})$ wrt. to $\omega_{i,0}^n$, since several local maxima exist, e.g. at multiples of $\omega_{i,0}^n$, see e.g. [6]. This problem is addressed by narrowing the search range based on prior knowledge of the domain, e.g. that the pitch of speech lies in the range

50-400Hz. A candidate estimate for $\omega_{i,0}^n$ is obtained by computing the autocorrelation function of $s_{i,t}^n - (\mathbf{f}_i^n)^\top \mathbf{s}_{i,t-1}^n$. Grid search is performed in the vicinity of the candidate. For each point in the grid we optimize $\mathbf{d}_i^n$:

$$
\mathbf{d}_{i,\mathbf{new}}^n = \left[ \sum_{t=2}^{\tau} \begin{bmatrix} (\mathbf{M}_{i,t-1}^n) & \hat{\mathbf{s}}_{i,t-1}^n (\mathbf{u}_{i,t}^n)^\top \\ \mathbf{u}_{i,t}^n (\hat{\mathbf{s}}_{i,t-1}^n)^\top & \mathbf{u}_{i,t}^n (\mathbf{u}_{i,t}^n)^\top \end{bmatrix} \right]^{-1} \sum_{t=2}^{\tau} \begin{bmatrix} \mathbf{m}_{i,t,t-1}^n \\ \hat{s}_{i,t}^n \mathbf{u}_{i,t}^n \end{bmatrix} \tag{4}
$$

At each step of the EM-algorithm, the parameters are normalized by enforcing $\|\mathbf{A}_i\| = 1$,

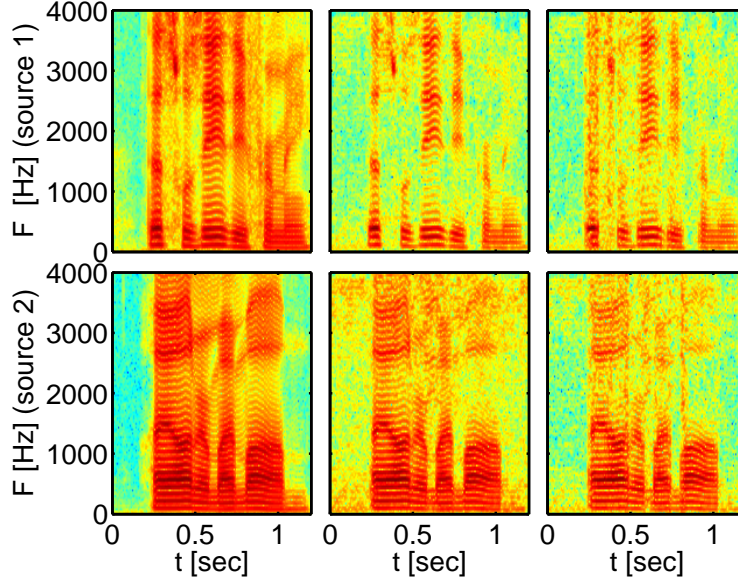

Figure 1: Amplitude spectrograms of the frequency range 0-4000Hz, from left to right: the true sources, the estimated sources and the re-synthesized source.

that is enforcing a unity norm on the filter coefficients related to source $i$.

## 4  Experiment I: BSS and pitch tracking in a noisy artificial mixture

The performance of a pitch detector can be evaluated using electro-laryngograph (EGG) recordings, which are obtained from electrodes placed on the neck, see [7]. In the following experiment, speech signals from the TIMIT [16] corpus is used for which the EGG signals were measured, kindly provided by the 'festvox' project (http://festvox.org).

Two male speech signals ($F_s = 16$kHz) were mixed through known mixing filters and degraded by additive white noise (SNR $\sim$20dB), constructing two observation signals. The pitches of the speech signals were overlapping. The filter coefficients (of $2 \times 2 = 4$ FIR filter impulse responses) were:

$$
\mathbf{A} = \begin{bmatrix} 1.00 & 0.35 & -0.20 & 0.00 & 0.00, & 0.00 & 0.00 & -0.50 & -0.30 & 0.20 \\ 0.00 & 0.00 & 0.70 & -0.20 & 0.15, & 1.30 & 0.60 & 0.30 & 0.00 & 0.00 \end{bmatrix}
$$

The signals were segmented into frames, $\tau = 320 \sim 20$ms, and the order of the AR-process was set to $p = 1$. The number of harmonics was limited to $K = 40$. The pitch grid search involved 30 re-estimations of $\mathbf{d}_i^n$. In figure 1 is shown the spectrograms of

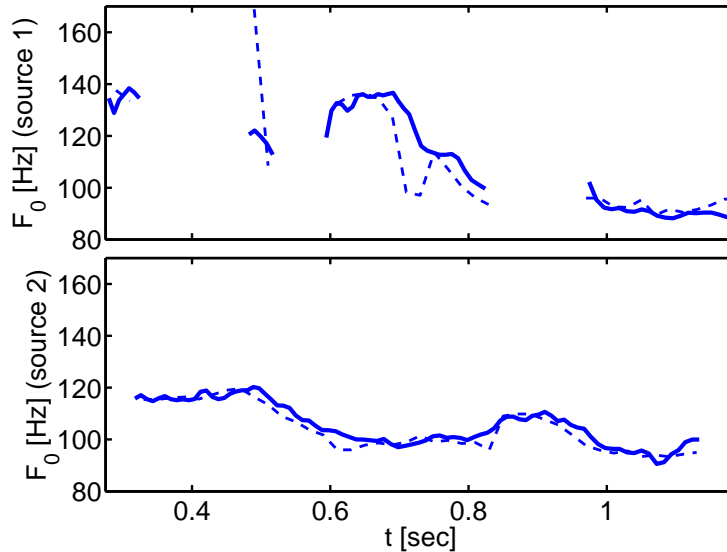

Figure 2: The estimated (dashed) and EGG-provided (solid) pitches as a function of time. The speech mixtures were artificially mixed from TIMIT utterances and white noise was added.

approximately 1 second of 1) the original sources, 2) the MAP source estimates and 3) the resynthesized sources (from the estimated model parameters). It is seen that the sources were well separated. Also, the re-synthesizations are almost indistinguishable from the source estimates. In figure 2, the estimated pitch of both speech signals are shown along with the pitch of the EGG measurements.[1] The voiced sections of the speech were manually preselected, this step is easily automated. The estimated pitches do follow the 'true' pitches as provided by the EGG. The smoothness of the estimates is further indicating the viability of the approach, as the pitch estimates are frame-local.

## 5   Experiment II: BSS and pitch tracking in a real mixture

The algorithm was further evaluated on real room recordings that were also used in [17].[2] Two male speakers synchronously count in English and Spanish ($F_s = 16$kHz). The mixtures were degraded with noise (SNR $\sim$20dB). The filter length, the frame length, the order of the AR-process and the number of harmonics were set to $L = 25$, $\tau = 320$, $p = 1$ and $K = 40$, respectively. Figure 3 shows the MAP source estimates and the re-synthesized sources. Features of speech such as amplitude modulation are clearly evident in estimates and re-synthesizations.[3] A listening test confirms: 1) the separation of the sources and 2) the good quality of the synthesized sources, reconfirming the applicability of the H+N model. Figure 4 displays the estimated pitches of the sources, where the voiced sections were manually preselected. Although, the 'true' pitch is unavailable in this experiment, the smoothness of the frame-local pitch-estimates is further support for the approach.

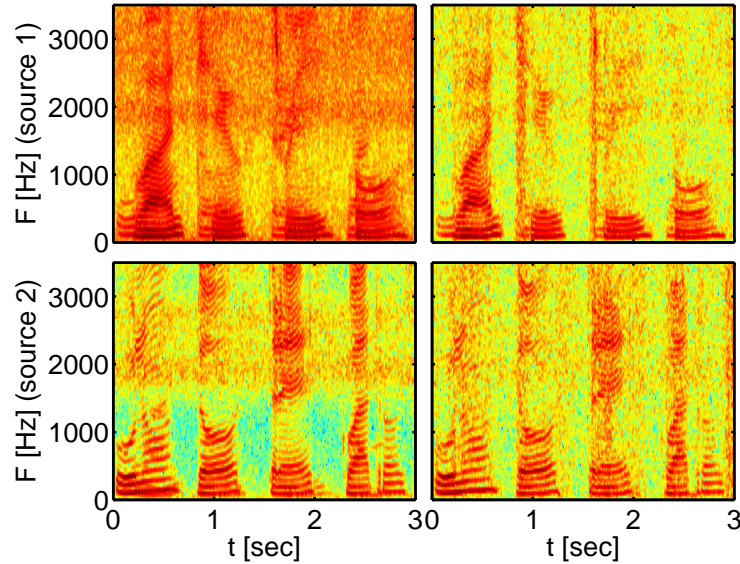

Figure 3: Spectrograms of the estimated (left) and re-synthesized sources (right) extracted from the 'one two ...' and 'uno dos ...' mixtures, source 1 and 2, respectively

## 6   Conclusion

It was shown that prior knowledge on speech signals and quasi-periodic signals in general can be integrated into a linear non-stationary state-space model. As a result, the simultaneous separation of the speech sources and estimation of their pitches could be achieved. It was demonstrated that the method could cope with noisy artificially mixed signals and real room mixtures. Future research concerns more realistic mixtures in terms of reverberation time and inclusion of further domain knowledge. It should be noted that the approach is computationally intensive, we are also investigating means for approximate inference and parameter estimation that would allow real time implementation.

### Acknowledgement

This work is supported by the Danish 'Oticon Fonden'.

## Footnotes

[1]The EGG data are themselves noisy measurements of the hypothesized 'truth'. Bandpass filtering was used for preprocessing.

[2]The mixtures were obtained from `http://inc2.ucsd.edu/~tewon/ica_cnl.html`.

[3]Note that the 'English' counter lowers the pitch throughout the sentence.

## References

[1] E. Weinstein, M. Feder and A.V. Oppenheim, Multi-channel signal separation by decorrelation, IEEE Trans. on speech and audio processing, vol. 1, no. 4, pp. 405-413,1993.

[2] Parra, L., Spence C., Convolutive blind separation of non-stationary sources. IEEE Trans. on speech and audio processing, vol. 5, pp. 320-327, 2000.

[3] Olsson, R. K., Hansen L. K., Probabilistic blind deconvolution of non-stationary source. Proc. EUSIPCO, 2004, *accepted*. Olsson R. K., Hansen L. K., Estimating the number of sources in a noisy convolutive mixture using BIC. International conference on independent component analysis 2004, *accepted*. Preprints may be obtained from `http://www.imm.dtu.dk/~rko/research.htm`.

[4] Gharbi, A.B.A., Salam, F., Blind separtion of independent sources in linear dynamical media. NOLTA, Hawaii, 1993. `http://www.egr.msu.edu/bsr/papers/blind_separation/nolta93.pdf`

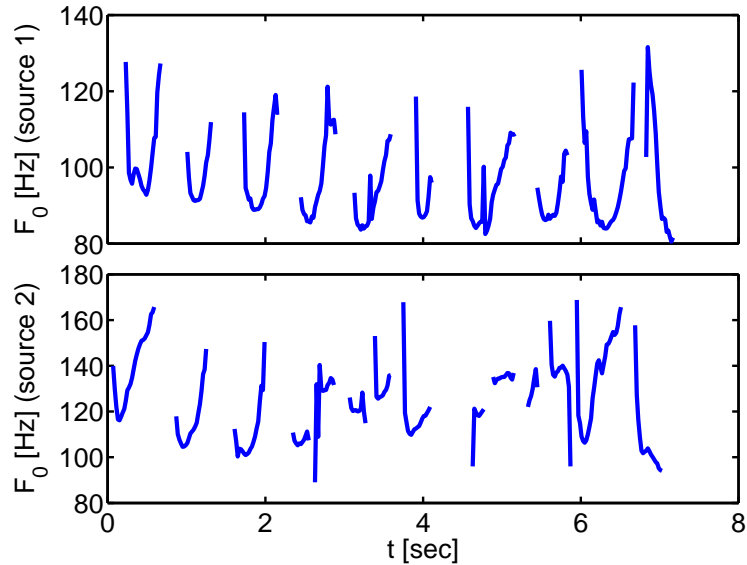

Figure 4: Pitch tracking in 'one two . . . '/'uno dos . . . ' mixtures.

[5] Zhang, L., Cichocki, A., Blind Deconvolution of dynamical systems: a state space appraoch, Journal of signal processing, vol. 4, no. 2, pp. 111-130, 2000.

[6] McAulay, R.J., Quateri. T.F., Speech analysis/synthesis based on a sinusoidal representation, IEEE Trans. on acoustics, speech and signal processing, vol. 34, no. 4, pp. 744-754, 1986.

[7] Parra, L., Jain U., Approximate Kalman filtering for the harmonic plus noise model. IEEE Workshop on applications of signal processing to audio and acoustics, pp. 75-78, 2001.

[8] Nakatani, T., Miyoshi, M., and Kinoshita, K., One microphone blind dereverberation based on quasi-periodicity of speech signals, Advances in Neural Information Processing Systems 16 (to appear), MIT Press, 2004.

[9] Hu, G. Wang, D., Monaural speech segregation based on pitch tracking and amplitude modulation, IEEE Trans. neural networks, in press, 2004.

[10] Anemüller, J., Kollmeier, B., Convolutive blind source separation of speech signals based on amplitude modulation decorrelation, Journal of the Acoustical Society of America, vol. 108, pp. 2630, 2000.

[11] A. P. Dempster, N. M. Laird, and Rubin D. B., Maximum liklihood from incomplete data via the EM algorithm, Journal of the Royal Statistical Society, vol. 39, pp. 1–38, 1977.

[12] Shumway, R.H., Stoffer, D.S., An approach to time series smoothing and forecasting using the EM algorithm. Journal of time series analysis, vol. 3, pp. 253-264. 1982.

[13] Moulines E., Cardoso J. F., Gassiat E., Maximum likelihood for blind separation and deconvolution of noisy signals using mixture models, ICASSP, vol. 5, pp. 3617-20, 1997.

[14] Cardoso, J.F., Snoussi, H. , Delabrouille, J., Patanchon, G., Blind separation of noisy Gaussian stationary sources. Application to cosmic microwave background imaging, Proc. EUSIPCO, pp 561-564, 2002.

[15] Roweis, S., Ghahramani, Z., A unifying review of linear Gaussian models. Neural Computation, vol. 11, pp. 305-345, 1999.

[16] Center for Speech Technology Research, University of Edinburgh, http://www.cstr.ed.ac.uk/

[17] Lee, T.-W., Bell, A.J., Orglmeister, R., Blind source separation of real world signals, Proc. IEEE international conference neural networks, pp 2129-2135, 1997.
